# Global Analytic Solution
# for Variational Bayesian Matrix Factorization

**Shinichi Nakajima**
Nikon Corporation
Tokyo, 140-8601, Japan
nakajima.s@nikon.co.jp

**Masashi Sugiyama**
Tokyo Institute of Technology
Tokyo 152-8552, Japan
sugi@cs.titech.ac.jp

**Ryota Tomioka**
The University of Tokyo
Tokyo 113-8685, Japan
tomioka@mist.i.u-tokyo.ac.jp

## Abstract

Bayesian methods of *matrix factorization* (MF) have been actively explored recently as promising alternatives to classical singular value decomposition. In this paper, we show that, despite the fact that the optimization problem is non-convex, the global optimal solution of *variational Bayesian* (VB) MF can be computed *analytically* by solving a *quartic* equation. This is highly advantageous over a popular VBMF algorithm based on iterated conditional modes since it can only find a local optimal solution after iterations. We further show that the global optimal solution of *empirical* VBMF (hyperparameters are also learned from data) can also be analytically computed. We illustrate the usefulness of our results through experiments.

## 1 Introduction

The problem of finding a low-rank approximation of a target matrix through *matrix factorization* (MF) attracted considerable attention recently since it can be used for various purposes such as *reduced rank regression* [19], *canonical correlation analysis* [8], *partial least-squares* [27, 21], *multi-class classification* [1], and *multi-task learning* [7, 29].

*Singular value decomposition* (SVD) is a classical method for MF, which gives the optimal low-rank approximation to the target matrix in terms of the squared error. Regularized variants of SVD have been studied for the *Frobenius-norm* penalty (i.e., singular values are regularized by the $\ell_2$-penalty) [17] or the *trace-norm* penalty (i.e., singular values are regularized by the $\ell_1$-penalty) [23]. Since the Frobenius-norm penalty does not automatically produce a low-rank solution, it should be combined with an explicit low-rank constraint, which is non-convex. In contrast, the trace-norm penalty tends to produce sparse solutions, so a low-rank solution can be obtained without explicit rank constraints. This implies that the optimization problem of trace-norm MF is still convex, and thus the global optimal solution can be obtained. Recently, optimization techniques for trace-norm MF have been extensively studied [20, 6, 12, 25].

Bayesian approaches to MF have also been actively explored. A *maximum a posteriori* (MAP) estimation, which computes the mode of the posterior distributions, was shown [23] to correspond to the $\ell_1$-MF when Gaussian priors are imposed on factorized matrices [22]. The *variational Bayesian* (VB) method [3, 5], which approximates the posterior distributions by factorized distributions, has also been applied to MF [13, 18]. The VB-based MF method (VBMF) was shown to perform well in experiments, and its theoretical properties have been investigated [15].

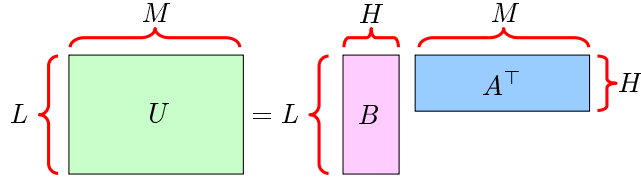

Figure 1: Matrix factorization model. $H \leq L \leq M$. $A = (\boldsymbol{a}_1, \ldots, \boldsymbol{a}_H)$ and $B = (\boldsymbol{b}_1, \ldots, \boldsymbol{b}_H)$.

However, the optimization problem of VBMF is non-convex. In practice, the VBMF solution is computed by the *iterated conditional modes* (ICM) [4, 5], where the mean and the covariance of the posterior distributions are iteratively updated until convergence [13, 18]. One may obtain a local optimal solution by the ICM algorithm, but many restarts would be necessary to find a good local optimum.

In this paper, we first show that, although the optimization problem is non-convex, the global optimal solution of VBMF can be computed *analytically* by solving a *quartic* equation. This is highly advantageous over the standard ICM algorithm since the global optimum can be found without any iterations and restarts. We next consider an *empirical* VB (EVB) scenario where the hyperparameters (prior variances) are also learned from data. Again, the optimization problem of EVBMF is non-convex, but we still show that the global optimal solution of EVBMF can be computed analytically. The usefulness of our results is demonstrated through experiments.

Recently, the global optimal solution of VBMF when the target matrix is square has been obtained in [15]. Thus, our contribution to VBMF can be regarded as an extension of the previous result to general rectangular matrices. On the other hand, for EVBMF, this is the first paper that gives the analytic global solution, to the best of our knowledge. The global analytic solution for EVBMF is shown to be highly useful in experiments.

## 2   Bayesian Matrix Factorization

In this section, we formulate the MF problem and review a variational Bayesian MF algorithm.

### 2.1   Formulation

The goal of MF is to approximate an unknown target matrix $U$ $(\in \mathbb{R}^{L \times M})$ from its $n$ observations

$$\mathcal{V}^n = \{V^{(i)} \in \mathbb{R}^{L \times M}\}_{i=1}^n.$$

We assume that $L \leq M$. If $L > M$, we may simply re-define the transpose $U^\top$ as $U$ so that $L \leq M$ holds. Thus this does not impose any restriction.

A key assumption of MF is that $U$ is a low-rank matrix. Let $H$ $(\leq L)$ be the rank of $U$. Then the matrix $U$ can be decomposed into the product of $A \in \mathbb{R}^{M \times H}$ and $B \in \mathbb{R}^{L \times H}$ as follows (see Figure 1):

$$U = BA^\top.$$

Assume that the observed matrix $V$ is subject to the following additive-noise model:

$$V = U + \mathcal{E},$$

where $\mathcal{E}$ $(\in \mathbb{R}^{L \times M})$ is a noise matrix. Each entry of $\mathcal{E}$ is assumed to independently follow the Gaussian distribution with mean zero and variance $\sigma^2$. Then, the likelihood $p(\mathcal{V}^n | A, B)$ is given by

$$p(\mathcal{V}^n | A, B) \propto \exp\left( -\frac{1}{2\sigma^2} \sum_{i=1}^n \|V^{(i)} - BA^\top\|_{\text{Fro}}^2 \right),$$

where $\| \cdot \|_{\text{Fro}}$ denotes the *Frobenius norm* of a matrix.

## 2.2 Variational Bayesian Matrix Factorization

We use the Gaussian priors on the parameters $A = (\boldsymbol{a}_1, \ldots, \boldsymbol{a}_H)$ and $B = (\boldsymbol{b}_1, \ldots, \boldsymbol{b}_H)$:

$$\phi(U) = \phi_A(A)\phi_B(B), \text{ where } \phi_A(A) \propto \exp\left(-\sum_{h=1}^{H} \frac{\|\boldsymbol{a}_h\|^2}{2c_{a_h}^2}\right) \text{ and } \phi_B(B) \propto \exp\left(-\sum_{h=1}^{H} \frac{\|\boldsymbol{b}_h\|^2}{2c_{b_h}^2}\right).$$

$c_{a_h}^2$ and $c_{b_h}^2$ are hyperparameters corresponding to the prior variance. Without loss of generality, we assume that the product $c_{a_h}c_{b_h}$ is non-increasing with respect to $h$.

Let $r(A, B|\mathcal{V}^n)$ be a *trial* distribution for $A$ and $B$, and let $F_{VB}$ be the *variational Bayes (VB) free energy* with respect to $r(A, B|\mathcal{V}^n)$:

$$F_{VB}(r|\mathcal{V}^n) = \left\langle \log \frac{r(A, B|\mathcal{V}^n)}{p(\mathcal{V}^n, A, B)} \right\rangle_{r(A,B|\mathcal{V}^n)},$$

where $\langle \cdot \rangle_p$ denotes the expectation over $p$.

The VB approach minimizes the VB free energy $F_{VB}(r|\mathcal{V}^n)$ with respect to the trial distribution $r(A, B|\mathcal{V}^n)$, by restricting the search space of $r(A, B|\mathcal{V}^n)$ so that the minimization is computationally tractable. Typically, dissolution of probabilistic dependency between entangled parameters ($A$ and $B$ in the case of MF) makes the calculation feasible:[1]

$$r(A, B|\mathcal{V}^n) = \prod_{h=1}^{H} r_{a_h}(\boldsymbol{a}_h|\mathcal{V}^n) r_{b_h}(\boldsymbol{b}_h|\mathcal{V}^n). \tag{1}$$

The resulting distribution is called the *VB posterior*. The VB solution $\widehat{U}^{VB}$ is given by the *VB posterior mean*:

$$\widehat{U}^{VB} = \langle BA^\top \rangle_{r(A,B|\mathcal{V}^n)}.$$

By applying the variational method to the VB free energy, we see that the VB posterior can be expressed as follows:

$$r(A, B|\mathcal{V}^n) = \prod_{h=1}^{H} \mathcal{N}_M(\boldsymbol{a}_h; \boldsymbol{\mu}_{a_h}, \Sigma_{a_h}) \mathcal{N}_L(\boldsymbol{b}_h; \boldsymbol{\mu}_{b_h}, \Sigma_{b_h}),$$

where $\mathcal{N}_d(\cdot; \boldsymbol{\mu}, \Sigma)$ denotes the $d$-dimensional Gaussian density with mean $\boldsymbol{\mu}$ and covariance matrix $\Sigma$. $\boldsymbol{\mu}_{a_h}, \boldsymbol{\mu}_{b_h}, \Sigma_{a_h}$, and $\Sigma_{b_h}$ satisfy

$$\boldsymbol{\mu}_{a_h} = \Sigma_{a_h} \Xi_h^\top \boldsymbol{\mu}_{b_h}, \quad \boldsymbol{\mu}_{b_h} = \Sigma_{b_h} \Xi_h \boldsymbol{\mu}_{a_h}, \quad \Sigma_{a_h} = \left(\frac{n\beta_h}{\sigma^2} + c_{a_h}^{-2}\right)^{-1} I_M, \quad \Sigma_{b_h} = \left(\frac{n\alpha_h}{\sigma^2} + c_{b_h}^{-2}\right)^{-1} I_L, \quad (2)$$

where $I_d$ denotes the $d$-dimensional identity matrix, and

$$\alpha_h = \|\boldsymbol{\mu}_{a_h}\|^2 + \text{tr}(\Sigma_{a_h}), \quad \beta_h = \|\boldsymbol{\mu}_{b_h}\|^2 + \text{tr}(\Sigma_{b_h}),$$

$$\Xi_h = \frac{n}{\sigma^2}\left(\overline{V} - \sum_{h' \neq h} \boldsymbol{\mu}_{b_{h'}} \boldsymbol{\mu}_{a_{h'}}^\top\right), \quad \overline{V} = \frac{1}{n}\sum_{i=1}^{n} V^{(i)}.$$

The *iterated conditional modes* (ICM) algorithm [4, 5] for VBMF (VB-ICM) iteratively updates $\boldsymbol{\mu}_{a_h}, \boldsymbol{\mu}_{b_h}, \Sigma_{a_h}$, and $\Sigma_{b_h}$ by Eq.(2) from some initial values until convergence [13, 18], allowing one to obtain a local optimal solution. Finally, an estimator of $U$ is computed as

$$\widehat{U}^{VB-ICM} = \sum_{h=1}^{H} \boldsymbol{\mu}_{b_h} \boldsymbol{\mu}_{a_h}^\top.$$

When the noise variance $\sigma^2$ is unknown, it may be estimated by the following re-estimation formula:

$$\sigma^2 = \frac{1}{\sigma^2 LM}\left(\frac{1}{n}\sum_{i=1}^{n}\left\|V^{(i)} - \sum_{h=1}^{H}\boldsymbol{\mu}_{b_h}\boldsymbol{\mu}_{a_h}^\top\right\|_{Fro}^2 + \sum_{h=1}^{H}\left(\alpha_h\beta_h - \|\boldsymbol{\mu}_{a_h}\|^2\|\boldsymbol{\mu}_{b_h}\|^2\right)\right),$$

which corresponds to the derivative of the VB free energy with respect to $\sigma^2$ set to zero (see Eq.(4) in Section 3). This can be incorporated in the ICM algorithm by updating $\sigma^2$ from some initial value by the above formula in every iteration of the ICM algorithm.

## 2.3 Empirical Variational Bayesian Matrix Factorization

In the VB framework, hyperparameters ($c_{a_h}^2$ and $c_{b_h}^2$ in the current setup) can also be learned from data by minimizing the VB free energy, which is called the *empirical VB (EVB)* method [5].

By setting the derivatives of the VB free energy with respect to $c_{a_h}^2$ and $c_{b_h}^2$ to zero, the following optimality condition can be obtained (see also Eq.(4) in Section 3):

$$c_{a_h}^2 = \alpha_h/M \quad \text{and} \quad c_{b_h}^2 = \beta_h/L. \tag{3}$$

The ICM algorithm for EVBMF (EVB-ICM) is to iteratively update $c_{a_h}^2$ and $c_{b_h}^2$ by Eq.(3), in addition to $\boldsymbol{\mu}_{a_h}$, $\boldsymbol{\mu}_{b_h}$, $\Sigma_{a_h}$, and $\Sigma_{b_h}$ by Eq.(2). Again, one may obtain a local optimal solution by this algorithm.

## 3 Analytic-form Expression of Global Optimal Solution of VBMF

In this section, we derive an analytic-form expression of the VBMF global solution.

The VB free energy can be explicitly expressed as follows.

$$F_{\mathrm{VB}}(r|\mathcal{V}^n) = \frac{nLM}{2}\log\sigma^2 + \sum_{h=1}^{H}\left(\frac{M}{2}\log c_{a_h}^2 - \frac{1}{2}\log|\Sigma_{a_h}| + \frac{\alpha_h}{2c_{a_h}^2} + \frac{L}{2}\log c_{b_h}^2 - \frac{1}{2}\log|\Sigma_{b_h}| + \frac{\beta_h}{2c_{b_h}^2}\right)$$

$$+ \frac{1}{2\sigma^2}\sum_{i=1}^{n}\left\|V^{(i)} - \sum_{h=1}^{H}\boldsymbol{\mu}_{b_h}\boldsymbol{\mu}_{a_h}^{\top}\right\|_{\mathrm{Fro}}^2 + \frac{n}{2\sigma^2}\sum_{h=1}^{H}\left(\alpha_h\beta_h - \|\boldsymbol{\mu}_{a_h}\|^2\|\boldsymbol{\mu}_{b_h}\|^2\right), \tag{4}$$

where $|\cdot|$ denotes the determinant of a matrix. We solve the following problem:

$$\begin{aligned}
\text{Given} \quad & (c_{a_h}^2, c_{b_h}^2) \in \mathbb{R}_{++}^2 \ (\forall h = 1, \dots, H), \ \sigma^2 \in \mathbb{R}_{++}, \\
\min \quad & F_{\mathrm{VB}}(\{\boldsymbol{\mu}_{a_h}, \boldsymbol{\mu}_{b_h}, \Sigma_{a_h}, \Sigma_{b_h}; h = 1, \dots, H\}) \\
\text{s.t.} \quad & \boldsymbol{\mu}_{a_h} \in \mathbb{R}^M, \ \boldsymbol{\mu}_{b_h} \in \mathbb{R}^L, \ \Sigma_{a_h} \in \mathbb{S}_{++}^M, \ \Sigma_{b_h} \in \mathbb{S}_{++}^L \ (\forall h = 1, \dots, H),
\end{aligned}$$

where $\mathbb{S}_{++}^d$ denotes the set of $d \times d$ symmetric positive-definite matrices. This is a non-convex optimization problem, but still we show that the global optimal solution can be analytically obtained.

Let $\gamma_h \ (\geq 0)$ be the $h$-th largest singular value of $\overline{V}$, and let $\boldsymbol{\omega}_{a_h}$ and $\boldsymbol{\omega}_{b_h}$ be the associated right and left singular vectors:[2]

$$\overline{V} = \sum_{h=1}^{L} \gamma_h \boldsymbol{\omega}_{b_h} \boldsymbol{\omega}_{a_h}^{\top}.$$

Let $\widehat{\gamma}_h$ be the *second* largest real solution of the following *quartic* equation with respect to $t$:

$$f_h(t) := t^4 + \xi_3 t^3 + \xi_2 t^2 + \xi_1 t + \xi_0 = 0, \tag{5}$$

where the coefficients are defined by

$$\xi_3 = \frac{(L-M)^2\gamma_h}{LM}, \quad \xi_2 = -\left(\xi_3\gamma_h + \frac{(L^2+M^2)\widehat{\eta}_h^2}{LM} + \frac{2\sigma^4}{n^2c_{a_h}^2c_{b_h}^2}\right), \quad \xi_1 = \xi_3\sqrt{\xi_0},$$

$$\xi_0 = \left(\widehat{\eta}_h^2 - \frac{\sigma^4}{n^2c_{a_h}^2c_{b_h}^2}\right)^2, \quad \widehat{\eta}_h^2 = \left(1 - \frac{\sigma^2 L}{n\gamma_h^2}\right)\left(1 - \frac{\sigma^2 M}{n\gamma_h^2}\right)\gamma_h^2.$$

Let

$$\widetilde{\gamma}_h = \sqrt{\frac{(L+M)\sigma^2}{2n} + \frac{\sigma^4}{2n^2c_{a_h}^2c_{b_h}^2} + \sqrt{\left(\frac{(L+M)\sigma^2}{2n} + \frac{\sigma^4}{2n^2c_{a_h}^2c_{b_h}^2}\right)^2 - \frac{LM\sigma^4}{n^2}}}. \tag{6}$$

Then we can *analytically* express the VBMF solution $\widehat{U}^{\mathrm{VB}}$ as in the following theorem.

**Theorem 1** *The global VB solution can be expressed as*

$$\widehat{U}^{\mathrm{VB}} = \sum_{h=1}^{H} \widehat{\gamma}_h^{\mathrm{VB}} \boldsymbol{\omega}_{b_h} \boldsymbol{\omega}_{a_h}^{\top}, \quad where \quad \widehat{\gamma}_h^{\mathrm{VB}} = \begin{cases} \widehat{\gamma}_h & \textit{if } \gamma_h > \widetilde{\gamma}_h, \\ 0 & \textit{otherwise.} \end{cases}$$

**Sketch of proof:** We first show that minimizing (4) amounts to a reweighed SVD and any minimizer is a stationary point. Then, by analyzing the stationary condition (2), we obtain an equation with respect to $\widehat{\gamma}_h$ as a necessary and sufficient condition to be a stationary point (note that its quadratic approximation gives bounds of the solution [15]). Its rigorous evaluation results in the quartic equation (5). Finally, we show that only the *second* largest solution of the quartic equation (5) lies within the bounds, which completes the proof. ∎

The coefficients of the quartic equation (5) are analytic, so $\widehat{\gamma}_h$ can also be obtained analytically[3], e.g., by *Ferrari's method* [9] (we omit the details due to lack of space). Therefore, the global VB solution can be analytically computed. This is a strong advantage over the standard ICM algorithm since many iterations and restarts would be necessary to find a good solution by ICM.

Based on the above result, the complete VB posterior can also be obtained analytically as follows.

**Corollary 2** *The VB posteriors are given by*

$$r_{\mathrm{A}}(A|\mathcal{V}^n) = \prod_{h=1}^{H} \mathcal{N}_M(\boldsymbol{a}_h; \boldsymbol{\mu}_{a_h}, \Sigma_{a_h}), \quad r_{\mathrm{B}}(B|\mathcal{V}^n) = \prod_{h=1}^{H} \mathcal{N}_M(\boldsymbol{b}_h; \boldsymbol{\mu}_{b_h}, \Sigma_{b_h}),$$

*where, for $\widehat{\gamma}_h^{\mathrm{VB}}$ being the solution given by Theorem 1,*

$$\boldsymbol{\mu}_{a_h} = \pm\sqrt{\widehat{\gamma}_h^{\mathrm{VB}}\widehat{\delta}_h} \cdot \boldsymbol{\omega}_{a_h}, \quad \boldsymbol{\mu}_{b_h} = \pm\sqrt{\widehat{\gamma}_h^{\mathrm{VB}}\widehat{\delta}_h^{-1}} \cdot \boldsymbol{\omega}_{b_h},$$

$$\Sigma_{a_h} = \left( \frac{-\left(n\widehat{\eta}_h^2 - \sigma^2(M-L)\right) + \sqrt{(n\widehat{\eta}_h^2 - \sigma^2(M-L))^2 + 4Mn\sigma^2\widehat{\eta}_h^2}}{2nM(\widehat{\gamma}_h^{\mathrm{VB}}\widehat{\delta}_h^{-1} + n^{-1}\sigma^2 c_{a_h}^{-2})} \right) I_M,$$

$$\Sigma_{b_h} = \left( \frac{-\left(n\widehat{\eta}_h^2 + \sigma^2(M-L)\right) + \sqrt{(n\widehat{\eta}_h^2 + \sigma^2(M-L))^2 + 4Ln\sigma^2\widehat{\eta}_h^2}}{2nL(\widehat{\gamma}_h^{\mathrm{VB}}\widehat{\delta}_h + n^{-1}\sigma^2 c_{b_h}^{-2})} \right) I_L,$$

$$\widehat{\delta}_h = \frac{n(M-L)(\gamma_h - \widehat{\gamma}_h^{\mathrm{VB}}) + \sqrt{n^2(M-L)^2(\gamma_h - \widehat{\gamma}_h^{\mathrm{VB}})^2 + \frac{4\sigma^4 LM}{c_{a_h}^2 c_{b_h}^2}}}{2\sigma^2 M c_{a_h}^{-2}},$$

$$\widehat{\eta}_h^2 = \begin{cases} \eta_h^2 & \textit{if } \gamma_h > \widetilde{\gamma}_h, \\ \frac{\sigma^2}{nc_{a_h}c_{b_h}} & \textit{otherwise.} \end{cases}$$

When the noise variance $\sigma^2$ is unknown, one may use the minimizer of the VB free energy with respect to $\sigma^2$ as its estimate. In practice, this single-parameter minimization may be carried out numerically based on Eq.(4) and Corollary 2.

## 4 Analytic-form Expression of Global Optimal Solution of Empirical VBMF

In this section, we solve the following problem to obtain the EVBMF global solution:

Given $\sigma^2 \in \mathbb{R}_{++}$,

min $F_{\mathrm{VB}}(\{\boldsymbol{\mu}_{a_h}, \boldsymbol{\mu}_{b_h}, \Sigma_{a_h}, \Sigma_{b_h}, c_{a_h}^2, c_{b_h}^2; h = 1, \ldots, H\})$

s.t. $\boldsymbol{\mu}_{a_h} \in \mathbb{R}^M, \boldsymbol{\mu}_{b_h} \in \mathbb{R}^L, \Sigma_{a_h} \in \mathbb{S}_{++}^M, \Sigma_{b_h} \in \mathbb{S}_{++}^L, (c_{a_h}^2, c_{b_h}^2) \in \mathbb{R}_{++}^2 \; (\forall h = 1, \ldots, H)$,

where $\mathbb{R}_{++}^d$ denotes the set of the $d$-dimensional vectors with positive elements. We show that, although this is again a non-convex optimization problem, the global optimal solution can be obtained analytically. We can observe the invariance of the VB free energy (4) under the transform

$$\left\{ (\boldsymbol{\mu}_{a_h}, \boldsymbol{\mu}_{b_h}, \Sigma_{a_h}, \Sigma_{b_h}, c_{a_h}^2, c_{b_h}^2) \right\} \to \left\{ (s_h\boldsymbol{\mu}_{a_h}, s_h^{-1}\boldsymbol{\mu}_{b_h}, s_h^2\Sigma_{a_h}, s_h^{-2}\Sigma_{b_h}, s_h^2 c_{a_h}^2, s_h^{-2} c_{b_h}^2) \right\}$$

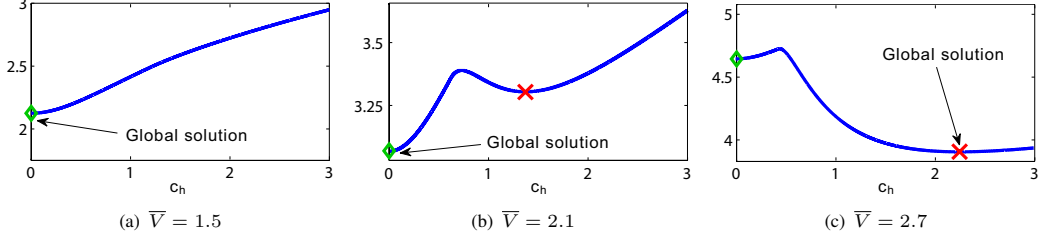

Figure 2: Profiles of the VB free energy (4) when $L = M = H = 1$, $n = 1$, and $\sigma^2 = 1$ for observations $\overline{V} = 1.5$, 2.1, and 2.7. (a) When $\overline{V} = 1.5 < 2 = \underline{\gamma}_h$, the VB free energy is monotone increasing and thus the global solution is given by $c_h \to 0$. (b) When $\overline{V} = 2.1 > 2 = \underline{\gamma}_h$, a local minimum exists at $c_h = \check{c}_h \approx 1.37$, but $\Delta_h \approx 0.12 > 0$ so $c_h \to 0$ is still the global solution. (c) When $\overline{V} = 2.7 > 2 = \underline{\gamma}_h$, $\Delta_h \approx -0.74 \leq 0$ and thus the minimizer at $c_h = \check{c}_h \approx 2.26$ is the global solution.

for any $\{s_h \neq 0; h = 1, \ldots, H\}$. Accordingly, we fix the ratios to $c_{a_h}/c_{b_h} = S > 0$, and refer to $c_h := c_{a_h} c_{b_h}$ also as a hyperparameter.

Let

$$\check{c}_h^2 = \frac{1}{2LM}\left(\gamma_h^2 - \frac{(L+M)\sigma^2}{n} + \sqrt{\left(\gamma_h^2 - \frac{(L+M)\sigma^2}{n}\right)^2 - \frac{4LM\sigma^4}{n^2}}\right), \qquad (7)$$

$$\underline{\gamma}_h = (\sqrt{L} + \sqrt{M})\sigma/\sqrt{n}.$$

Then, we have the following lemma:

**Lemma 3** *If $\gamma_h \geq \underline{\gamma}_h$, the VB free energy function (4) can have two local minima, namely, $c_h \to 0$ and $c_h = \check{c}_h$. Otherwise, $c_h \to 0$ is the only local minimum of the VB free energy.*

**Sketch of proof:** Analyzing the region where $c_h$ is so small that the VB solution given $c_h$ is $\widehat{\gamma}_h = 0$, we find a local minimum $c_h \to 0$. Combining the stationary conditions (2) and (3), we derive a quadratic equation with respect to $c_h^2$ whose *larger* solution is given by Eq.(7). Showing that the *smaller* solution corresponds to saddle points completes the proof. ∎

Figure 2 shows the profiles of the VB free energy (4) when $L = M = H = 1$, $n = 1$, and $\sigma^2 = 1$ for observations $\overline{V} = 1.5$, 2.1, and 2.7. As illustrated, depending on the value of $\overline{V}$, either $c_h \to 0$ or $c_h = \check{c}_h$ is the global solution.

Let

$$\Delta_h := M \log\left(\frac{n\gamma_h}{M\sigma^2}\check{\gamma}_h^{\mathrm{VB}} + 1\right) + L \log\left(\frac{n\gamma_h}{L\sigma^2}\check{\gamma}_h^{\mathrm{VB}} + 1\right) + \frac{n}{\sigma^2}\left(-2\gamma_h\check{\gamma}_h^{\mathrm{VB}} + LM\check{c}_h^2\right), \quad (8)$$

where $\check{\gamma}_h^{\mathrm{VB}}$ is the VB solution for $c_h = \check{c}_h$. We can show that the sign of $\Delta_h$ corresponds to that of the difference of the VB free energy at $c_h = \check{c}_h$ and $c_h \to 0$. Then, we have the following theorem and corollary.

**Theorem 4** *The hyperparameter $\widehat{c}_h$ that globally minimizes the VB free energy function (4) is given by $\widehat{c}_h = \check{c}_h$ if $\gamma_h > \underline{\gamma}_h$ and $\Delta_h \leq 0$. Otherwise $\widehat{c}_h \to 0$.*

**Corollary 5** *The global EVB solution can be expressed as*

$$\widehat{U}^{\mathrm{EVB}} = \sum_{h=1}^{H}\widehat{\gamma}_h^{\mathrm{EVB}}\boldsymbol{\omega}_{b_h}\boldsymbol{\omega}_{a_h}^{\top}, \quad \text{where} \quad \widehat{\gamma}_h^{\mathrm{EVB}} := \begin{cases} \check{\gamma}_h^{\mathrm{VB}} & \text{if } \gamma_h > \underline{\gamma}_h \text{ and } \Delta_h \leq 0, \\ 0 & \text{otherwise.} \end{cases}$$

Since the optimal hyperparameter value $\widehat{c}_h$ can be expressed in a closed-form, the global EVB solution can also be computed analytically using the result given in Section 3. This is again a strong advantage over the standard ICM algorithm since ICM would require many iterations and restarts to find a good solution.

# 5  Experiments

In this section, we experimentally evaluate the usefulness of our analytic-form solutions using artificial and benchmark datasets. The MATLAB$^{\circledR}$ code will be available at [14].

## 5.1  Artificial Dataset

We randomly created a *true* matrix $V^* = \sum_{h=1}^{H^*} \boldsymbol{b}_h^* \boldsymbol{a}_h^{*\top}$ with $L = 30$, $M = 100$, and $H^* = 10$, where every element of $\{\boldsymbol{a}_h, \boldsymbol{b}_h\}$ was drawn independently from the standard Gaussian distribution. We set $n = 1$, and an observation matrix $V$ was created by adding independent Gaussian noise with variance $\sigma^2 = 1$ to each element. We used the full-rank model, i.e., $H = L = 30$. The noise variance $\sigma^2$ was assumed to be unknown, and estimated from data (see Section 2.2 and Section 3).

We first investigate the learning curve of the VB free energy over EVB-ICM iterations. We created the initial values of the EVB-ICM algorithm as follows: $\boldsymbol{\mu}_{a_h}$ and $\boldsymbol{\mu}_{b_h}$ were set to randomly created orthonormal vectors, $\Sigma_{a_h}$ and $\Sigma_{b_h}$ were set to identity matrices multiplied by scalars $\sigma_{a_h}^2$ and $\sigma_{b_h}^2$, respectively. $\sigma_{a_h}^2$ and $\sigma_{b_h}^2$ as well as the noise variance $\sigma^2$ were drawn from the $\chi^2$-distribution with degree-of-freedom one. 10 learning curves of the VB free energy were plotted in Figures 3(a). The value of the VB free energy of the global solution computed by our analytic-form solution was also plotted in the graph by the dashed line. The graph shows that the EVB-ICM algorithm reduces the VB free energy reasonably well over iterations. However, for this artificial dataset, the convergence speed was quite slow once in 10 runs, which was actually trapped in a local minimum.

Next, we compare the computation time. Figure 3(b) shows the computation time of EVB-ICM over iterations and our analytic form-solution. The computation time of EVB-ICM grows almost linearly with respect to the number of iterations, and it took 86.6 [sec] for 100 iterations on average. On the other hand, the computation of our analytic-form solution took only 0.055 [sec] on average, including the single-parameter search for $\sigma^2$. Thus, our method provides the reduction of computation time in 4 orders of magnitude, with better accuracy as a minimizer of the VB free energy.

Next, we investigate the generalization error of the global analytic solutions of VB and EVB, measured by $G = \|\widehat{U} - V^*\|_{\mathrm{Fro}}^2 / (LM)$. Figure 3(c) shows the mean and error bars (min and max) over 10 runs for VB with various hyperparameter values and EVB. A single hyperparameter value was commonly used (i.e., $c_1 = \cdots = c_H$) in VB, while each hyperparameter $c_h$ was separately optimized in EVB. The result shows that EVB gives slightly lower generalization errors than VB with the best common hyperparameter. Thus, automatic hyperparameter selection of EVB works quite well.

Figure 3(d) shows the hyperparameter values chosen in EVB sorted in the decreasing order. This shows that, for all 10 runs, $c_h$ is positive for $h \leq H^* (= 10)$ and zero for $h > H^*$. This implies that the effect of *automatic relevance determination* [16, 5] works excellently for this artificial dataset.

## 5.2  Benchmark Dataset

MF can be used for *canonical correlation analysis* (CCA) [8] and *reduced rank regression* (RRR) [19] with appropriately pre-whitened data. Here, we solve these tasks by VBMF and evaluate the performance using the *concrete slump test* dataset [28] available from the UCI repository [2].

The experimental results are depicted in Figure 4, which is in the same format as Figure 3. The results showed that similar trends to the artificial dataset can still be observed for the CCA task with the benchmark dataset (the RRR results are similar and thus omitted from the figure). Overall, the proposed global analytic solution is shown to be a useful alternative to the popular ICM algorithm.

# 6  Discussion and Conclusion

Overcoming the non-convexity of VB methods has been one of the important challenges in the Bayesian machine learning community, since it sometimes prevented us from applying the VB methods to highly complex real-world problems. In this paper, we focused on the MF problem with no missing entry, and showed that this weakness could be overcome by computing the global optimal solution *analytically*. We further derived the global optimal solution analytically for the EVBMF

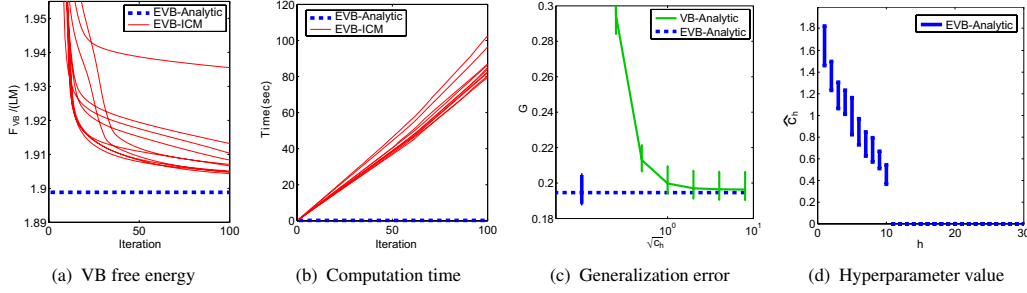

Figure 3: Experimental results for artificial dataset.

(a) VB free energy      (b) Computation time      (c) Generalization error      (d) Hyperparameter value

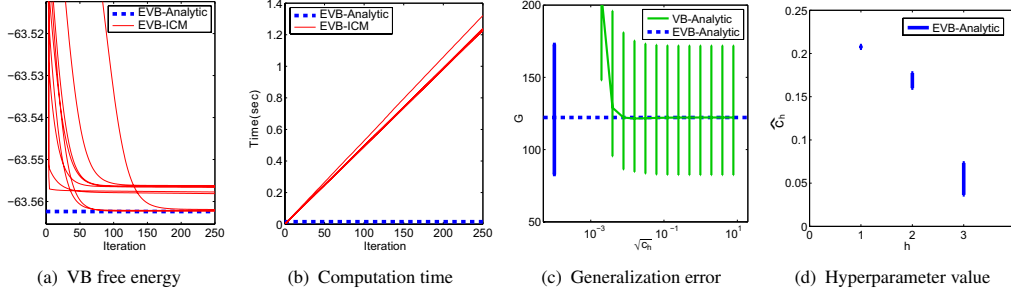

(a) VB free energy      (b) Computation time      (c) Generalization error      (d) Hyperparameter value

Figure 4: Experimental results of CCA for the *concrete slump test* dataset.

method, where hyperparameters are also optimized based on data samples. Since no hand-tuning parameter remains in EVBMF, our analytic-form solution is practically useful and computationally highly efficient. Numerical experiments showed that the proposed approach is promising.

When $c_{a_h} c_{b_h} \to \infty$, the priors get (almost) flat and the quartic equation (5) is factorized as

$$\lim_{c_{a_h} c_{b_h} \to \infty} f_h(t) = \left(t + \frac{M}{L}\left(1 - \frac{\sigma^2 L}{n\gamma_h^2}\right)\gamma_h\right)\left(t + \left(1 - \frac{\sigma^2 M}{n\gamma_h^2}\right)\gamma_h\right)\left(t - \left(1 - \frac{\sigma^2 M}{n\gamma_h^2}\right)\gamma_h\right)\left(t - \frac{M}{L}\left(1 - \frac{\sigma^2 L}{n\gamma_h^2}\right)\gamma_h\right) = 0.$$

Theorem 1 states that its *second* largest solution gives the VB estimator for $\gamma_h > \lim_{c_{a_h} c_{b_h} \to \infty} \widetilde{\gamma}_h = \sqrt{M\sigma^2/n}$. Thus we have

$$\lim_{c_{a_h} c_{b_h} \to \infty} \widehat{\gamma}_h^{\mathrm{VB}} = \max\left(0, \left(1 - \frac{M\sigma^2}{n\gamma_h^2}\right)\right)\gamma_h.$$

This is the *positive-part James-Stein* (PJS) shrinkage estimator [10], operated on each singular component separately, and this coincides with the upper-bound derived in [15] for arbitrary $c_{a_h} c_{b_h} > 0$. The counter-intuitive fact—a shrinkage is observed even in the limit of flat priors—can be explained by strong non-uniformity of the *volume element of the Fisher metric*, i.e., the *Jeffreys* prior [11], in the parameter space. We call this effect *model-induced regularization* (MIR), because it is induced not by priors but by structure of model likelihood functions. MIR was shown to generally appear in Bayesian estimation when the model is *non-identifiable* (i.e., the mapping between parameters and distribution functions is not one-to-one) and the parameters are integrated out *at least partially* [26]. Thus, it never appears in MAP estimation [15]. The probabilistic PCA can be seen as an example of MF, where $A$ and $B$ correspond to latent variables and principal axes, respectively [24]. The MIR effect is observed in its analytic solution when $A$ is integrated out and $B$ is estimated to be the maximizer of the marginal likelihood.

Our results fully made use of the assumptions that the likelihood and priors are both spherical Gaussian, the VB posterior is column-wise independent, and there exists no missing entry. They were necessary to solve the free energy minimization problem as a reweighted SVD. An important future work is to obtain the analytic global solution under milder assumptions. This will enable us to handle more challenging problems such as missing entry prediction [23, 20, 6, 13, 18, 22, 12, 25].

**Acknowledgments**

The authors appreciate comments by anonymous reviewers, which helped improve our earlier manuscript and suggested promising directions for future work. MS thanks the support from the FIRST program. RT was partially supported by MEXT Kakenhi 22700138.

## Footnotes

[1]Although a weaker constraint, $r(A, B|\mathcal{V}^n) = r_A(A|\mathcal{V}^n)r_B(B|\mathcal{V}^n)$, is sufficient to derive a tractable iterative algorithm [13], we assume the stronger one (1) used in [18], which makes our theoretical analysis tractable.

[2] In our analysis, we assume that $\overline{V}$ has no missing entry, and its singular value decomposition (SVD) is easily obtained. Therefore, our results cannot be directly applied to missing entry prediction.

[3]In practice, one may solve the quartic equation numerically, e.g., by the 'roots' function in MATLAB®.

# References

[1] Y. Amit, M. Fink, N. Srebro, and S. Ullman. Uncovering shared structures in multiclass classification. In *Proceedings of International Conference on Machine Learning*, pages 17–24, 2007.

[2] A. Asuncion and D.J. Newman. UCI machine learning repository, 2007.

[3] H. Attias. Inferring parameters and structure of latent variable models by variational Bayes. In *Proceedings of the Fifteenth Conference Annual Conference on Uncertainty in Artificial Intelligence (UAI-99)*, pages 21–30, San Francisco, CA, 1999. Morgan Kaufmann.

[4] J. Besag. On the Statistical Analysis of Dirty Pictures. *J. Royal Stat. Soc. B*, 48:259–302, 1986.

[5] C. M. Bishop. *Pattern Recognition and Machine Learning*. Springer, New York, NY, USA, 2006.

[6] J.-F. Cai, E. J. Candes, and Z. Shen. A singular value thresholding algorithm for matrix completion. *SIAM Journal on Optimization*, 20(4):1956–1982, 2008.

[7] O. Chapelle and Z. Harchaoui. A Machine Learning Approach to Conjoint Analysis. In *Advances in neural information processing systems*, volume 17, pages 257–264, 2005.

[8] D. R. Hardoon, S. R. Szedmak, and J. R. Shawe-Taylor. Canonical correlation analysis: An overview with application to learning methods. *Neural Computation*, 16(12):2639–2664, 2004.

[9] M. Hazewinkel, editor. *Encyclopaedia of Mathematics*. Springer, 2002.

[10] W. James and C. Stein. Estimation with quadratic loss. In *Proceedings of the 4th Berkeley Symposium on Mathematical Statistics and Probability*, volume 1, pages 361–379. University of California Press, 1961.

[11] H. Jeffreys. An Invariant Form for the Prior Probability in Estimation Problems. In *Proceedings of the Royal Society of London. Series A*, volume 186, pages 453–461, 1946.

[12] S. Ji and J. Ye. An accelerated gradient method for trace norm minimization. In *Proceedings of International Conference on Machine Learning*, pages 457–464, 2009.

[13] Y. J. Lim and T. W. Teh. Variational Bayesian Approach to Movie Rating Prediction. In *Proceedings of KDD Cup and Workshop*, 2007.

[14] S. Nakajima. Matlab Code for VBMF, http://sites.google.com/site/shinnkj23/, 2010.

[15] S. Nakajima and M. Sugiyama. Implicit regularization in variational Bayesian matrix factorization. In *Proceedings of 27th International Conference on Machine Learning (ICML2010)*, 2010.

[16] R. M. Neal. *Bayesian Learning for Neural Networks*. Springer, 1996.

[17] A. Paterek. Improving Regularized Singular Value Decomposition for Collaborative Filtering. In *Proceedings of KDD Cup and Workshop*, 2007.

[18] T. Raiko, A. Ilin, and J. Karhunen. Principal Component Analysis for Large Sale Problems with Lots of Missing Values. In *Proc. of ECML*, volume 4701, pages 691–698, 2007.

[19] G. R. Reinsel and R. P. Velu. *Multivariate reduced-rank Regression: Theory and Applications*. Springer, New York, 1998.

[20] J. D. M. Rennie and N. Srebro. Fast maximum margin matrix factorization for collaborative prediction. In *Proceedings of the 22nd International Conference on Machine learning*, pages 713–719, 2005.

[21] R. Rosipal and N. Krämer. Overview and recent advances in partial least squares. In *Subspace, Latent Structure and Feature Selection Techniques*, volume 3940, pages 34–51. Springer, 2006.

[22] R. Salakhutdinov and A. Mnih. Probabilistic matrix factorization. In J.C. Platt, D. Koller, Y. Singer, and S. Roweis, editors, *Advances in Neural Information Processing Systems 20*, pages 1257–1264, 2008.

[23] N. Srebro, J. Rennie, and T. Jaakkola. Maximum Margin Matrix Factorization. In *Advances in NIPS*, volume 17, 2005.

[24] M.E. Tipping and C.M. Bishop. Probabilistic Principal Component Analysis. *Journal of the Royal Statistical Society: Series B*, 61(3):611–622, 1999.

[25] R. Tomioka, T. Suzuki, M. Sugiyama, and H. Kashima. An efficient and general augmented Lagrangian algorithm for learning low-rank matrices. In *Proceedings of International Conference on Machine Learning*, 2010.

[26] S. Watanabe. *Algebraic Geometry and Statistical Learning*. Cambridge University Press, Cambridge, UK, 2009.

[27] K. J. Worsley, J-B. Poline, K. J. Friston, and A. C. Evanss. Characterizing the Response of PET and fMRI Data Using Multivariate Linear Models. *NeuroImage*, 6(4):305–319, 1997.

[28] I-Cheng Yeh. Modeling slump flow of concrete using second-order regressions and artificial neural networks. *Cement and Concrete Composites*, 29(6):474–480, 2007.

[29] K. Yu, V. Tresp, and A. Schwaighofer. Learning Gaussian Processes from Multiple Tasks. In *Proc. of ICML*, page 1019, 2005.

